# Accelerated Gradient Methods for Stochastic Optimization and Online Learning

**Chonghai Hu[♯†], James T. Kwok[♯], Weike Pan[♯]**
♯ Department of Computer Science and Engineering
Hong Kong University of Science and Technology
Clear Water Bay, Kowloon, Hong Kong
† Department of Mathematics, Zhejiang University
Hangzhou, China
hino.hu@gmail.com, {jamesk,weikep}@cse.ust.hk

## Abstract

Regularized risk minimization often involves non-smooth optimization, either because of the loss function (e.g., hinge loss) or the regularizer (e.g., $\ell_1$-regularizer). Gradient methods, though highly scalable and easy to implement, are known to converge slowly. In this paper, we develop a novel accelerated gradient method for stochastic optimization while still preserving their computational simplicity and scalability. The proposed algorithm, called SAGE (Stochastic Accelerated GradiEnt), exhibits fast convergence rates on stochastic composite optimization with convex or strongly convex objectives. Experimental results show that SAGE is faster than recent (sub)gradient methods including FOLOS, SMIDAS and SCD. Moreover, SAGE can also be extended for online learning, resulting in a simple algorithm but with the best regret bounds currently known for these problems.

## 1 Introduction

Risk minimization is at the heart of many machine learning algorithms. Given a class of models parameterized by $w$ and a loss function $\ell(\cdot, \cdot)$, the goal is to minimize $\mathbb{E}_{XY}[\ell(w; X, Y)]$ w.r.t. $w$, where the expectation is over the joint distribution of input $X$ and output $Y$. However, since the joint distribution is typically unknown in practice, a surrogate problem is to replace the expectation by its empirical average on a training sample $\{(x_1, y_1), \ldots, (x_m, y_m)\}$. Moreover, a regularizer $\Omega(\cdot)$ is often added for well-posedness. This leads to the minimization of the regularized risk

$$\min_w \quad \frac{1}{m} \sum_{i=1}^m \ell(w; x_i, y_i) + \lambda\Omega(w), \tag{1}$$

where $\lambda$ is a regularization parameter. In optimization terminology, the deterministic optimization problem in (1) can be considered as a sample average approximation (SAA) of the corresponding stochastic optimization problem:

$$\min_w \quad \mathbb{E}_{XY}[\ell(w; X, Y)] + \lambda\Omega(w). \tag{2}$$

Since both $\ell(\cdot, \cdot)$ and $\Omega(\cdot)$ are typically convex, (1) is a convex optimization problem which can be conveniently solved even with standard off-the-shelf optimization packages.

However, with the proliferation of data-intensive applications in the text and web domains, data sets with millions or trillions of samples are nowadays not uncommon. Hence, off-the-shelf optimization solvers are too slow to be used. Indeed, even tailor-made softwares for specific models, such as the sequential minimization optimization (SMO) method for the SVM, have superlinear computational

complexities and thus are not feasible for large data sets. In light of this, the use of stochastic methods have recently drawn a lot of interest and many of these are highly successful. Most are based on (variants of) the stochastic gradient descent (SGD). Examples include Pegasos [1], SGD-QN [2], FOLOS [3], and stochastic coordinate descent (SCD) [4]. The main advantages of these methods are that they are simple to implement, have low per-iteration complexity, and can scale up to large data sets. Their runtime is independent of, or even decrease with, the number of training samples [5, 6]. On the other hand, because of their simplicity, these methods have a slow convergence rate, and thus may require a large number of iterations.

While standard gradient schemes have a slow convergence rate, they can often be "accelerated". This stems from the pioneering work of Nesterov in 1983 [7], which is a deterministic algorithm for smooth optimization. Recently, it is also extended for composite optimization, where the objective has a smooth component and a non-smooth component [8, 9]. This is particularly relevant to machine learning since the loss $\ell$ and regularizer $\Omega$ in (2) may be non-smooth. Examples include loss functions such as the commonly-used hinge loss used in the SVM, and regularizers such as the popular $\ell_1$ penalty in Lasso [10], and basis pursuit. These accelerated gradient methods have also been successfully applied in the optimization problems of multiple kernel learning [11] and trace norm minimization [12]. Very recently, Lan [13] made an initial attempt to further extend this for stochastic composite optimization, and obtained the convergence rate of

$$\mathcal{O}\left(L/N^2 + (M + \sigma)/\sqrt{N}\right). \tag{3}$$

Here, $N$ is the number of iterations performed by the algorithm, $L$ is the Lipschitz parameter of the gradient of the smooth term in the objective, $M$ is the Lipschitz parameter of the nonsmooth term, and $\sigma$ is the variance of the stochastic subgradient. Moreover, note that the first term of (3) is related to the smooth component in the objective while the second term is related to the non-smooth component. Complexity results [14, 13] show that (3) is the optimal convergence rate for any iterative algorithm solving stochastic (general) convex composite optimization.

However, as pointed out in [15], a very useful property that can improve the convergence rates in machine learning optimization problems is strong convexity. For example, (2) can be strongly convex either because of the strong convexity of $\ell$ (e.g., log loss, square loss) or $\Omega$ (e.g., $\ell_2$ regularization). On the other hand, [13] is more interested in general convex optimization problems and so strong convexity is not utilized. Moreover, though theoretically interesting, [13] may be of limited practical use as (1) the stepsize in its update rule depends on the often unknown $\sigma$; and (2) the number of iterations performed by the algorithm has to be fixed in advance.

Inspired by the successes of Nesterov's method, we develop in this paper a novel accelerated subgradient scheme for stochastic composite optimization. It achieves the optimal convergence rate of $\mathcal{O}\left(L/N^2 + \sigma/\sqrt{N}\right)$ for general convex objectives, and $\mathcal{O}\left((L+\mu)/N^2 + \sigma\mu^{-1}/N\right)$ for $\mu$-strongly convex objectives. Moreover, its per-iteration complexity is almost as low as that for standard (sub)gradient methods. Finally, we also extend the accelerated gradient scheme to online learning. We obtain $\mathcal{O}(\sqrt{N})$ regret for general convex problems and $\mathcal{O}(\log N)$ regret for strongly convex problems, which are the best regret bounds currently known for these problems.

## 2 Setting and Mathematical Background

First, we recapitulate a few notions in convex analysis.

**(Lipschitz continuity)** A function $f(x)$ is $L$-Lipschitz if $\|f(x) - f(y)\| \leq L\|x - y\|$.

**Lemma 1.** *[14] The gradient of a differentiable function $f(x)$ is Lipschitz continuous with Lipschitz parameter $L$ if, for any $x$ and $y$,*

$$f(y) \leq f(x) + \langle \nabla f(x), y - x \rangle + \frac{L}{2}\|x - y\|^2. \tag{4}$$

**(Strong convexity)** A function $\phi(x)$ is $\mu$-strongly convex if $\phi(y) \geq \phi(x) + \langle g(x), y - x \rangle + \frac{\mu}{2}\|y - x\|^2$ for any $x, y$ and subgradient $g(x) \in \partial\phi(x)$.

**Lemma 2.** *[14] Let $\phi(x)$ be $\mu$-strongly convex and $x^* = \arg\min_x \phi(x)$. Then, for any $x$,*

$$\phi(x) \geq \phi(x^*) + \frac{\mu}{2}\|x - x^*\|^2. \tag{5}$$

We consider the following stochastic convex stochastic optimization problem, with a composite objective function

$$\min_x \{\phi(x) \equiv \mathbb{E}[F(x,\xi)] + \psi(x)\}, \tag{6}$$

where $\xi$ is a random vector, $f(x) \equiv \mathbb{E}[F(x,\xi)]$ is convex and differentiable, and $\psi(x)$ is convex but non-smooth. Clearly, this includes the optimization problem (2). Moreover, we assume that the gradient of $f(x)$ is $L$-Lipschitz and $\phi(x)$ is $\mu$-strongly convex (with $\mu \geq 0$). Note that when $\phi(x)$ is smooth ($\psi(x) = 0$), $\mu$ lower bounds the smallest eigenvalue of its Hessian.

Recall that in smooth optimization, the gradient update $x_{t+1} = x_t - \lambda \nabla f(x_t)$ on a function $f(x)$ can be seen as proximal regularization of the linearized $f$ at the current iterate $x_t$ [16]. In other words, $x_{t+1} = \arg\min_x (\langle \nabla f(x_t), x - x_t \rangle + \frac{1}{2\lambda}\|x - x_t\|^2)$. With the presence of a non-smooth component, we have the following more general notion.

**(Gradient mapping)** [8] In minimizing $f(x) + \psi(x)$, where $f$ is convex and differentiable and $\psi$ is convex and non-smooth,

$$x_{t+1} = \arg\min_x \left( \langle \nabla f(x), x - x_t \rangle + \frac{1}{2\lambda}\|x - x_t\|^2 + \psi(x) \right) \tag{7}$$

is called the generalized gradient update, and $\delta = \frac{1}{\lambda}(x_t - x_{t+1})$ is the (generalized) gradient mapping. Note that the quadratic approximation is made to the smooth component only. It can be shown that the gradient mapping is analogous to the gradient in smooth convex optimization [14, 8]. This is also a common construct used in recent stochastic subgradient methods [3, 17].

## 3   Accelerated Gradient Method for Stochastic Learning

Let $G(x_t, \xi_t) \equiv \nabla_x F(x, \xi_t)|_{x=x_t}$ be the stochastic gradient of $F(x, \xi_t)$. We assume that it is an unbiased estimator of the gradient $\nabla f(x)$, i.e., $\mathbb{E}_\xi[G(x,\xi)] = \nabla f(x)$. Algorithm 1 shows the proposed algorithm, which will be called SAGE (Stochastic Accelerated GradiEnt). It involves the updating of three sequences $\{x_t\}$, $\{y_t\}$ and $\{z_t\}$. Note that $y_t$ is the generalized gradient update, and $x_{t+1}$ is a convex combination of $y_t$ and $z_t$. The algorithm also maintains two parameter sequences $\{\alpha_t\}$ and $\{L_t\}$. We will see in Section 3.1 that different settings of these parameters lead to different convergence rates. Note that the only expensive step of Algorithm 1 is the computation of the generalized gradient update $y_t$, which is analogous to the subgradient computation in other subgradient-based methods. In general, its computational complexity depends on the structure of $\psi(x)$. As will be seen in Section 3.3, this can often be efficiently obtained in many regularized risk minimization problems.

---
**Algorithm 1** SAGE (Stochastic Accelerated GradiEnt).

---
Input: Sequences $\{L_t\}$ and $\{\alpha_t\}$.
Initialize: $y_{-1} = z_{-1} = 0$, $\alpha_0 = \lambda_0 = 1$. $L_0 = L + \mu$.
**for** $t = 0$ to $N$ **do**
    $x_t = (1 - \alpha_t)y_{t-1} + \alpha_t z_{t-1}$.
    $y_t = \arg\min_x \left\{ \langle G(x_t, \xi_t), x - x_t \rangle + \frac{L_t}{2}\|x - x_t\|^2 + \psi(x) \right\}$.
    $z_t = z_{t-1} - (L_t\alpha_t + \mu)^{-1}[L_t(x_t - y_t) + \mu(z_{t-1} - x_t)]$.
**end for**
Output $y_N$.

---

### 3.1   Convergence Analysis

Define $\Delta_t \equiv G(x_t, \xi_t) - \nabla f(x_t)$. Because of the unbiasedness of $G(x_t, \xi_t)$, $\mathbb{E}_{\xi_t}[\Delta_t] = 0$. In the following, we will show that the value of $\phi(y_t) - \phi(x)$ can be related to that of $\phi(y_{t-1}) - \phi(x)$ for any $x$. Let $\delta_t \equiv L_t(x_t - y_t)$ be the gradient mapping involved in updating $y_t$. First, we introduce the following lemma.

**Lemma 3.** *For $t \geq 0$, $\phi(x)$ is quadratically bounded from below as*

$$\phi(x) \geq \phi(y_t) + \langle \delta_t, x - x_t \rangle + \frac{\mu}{2}\|x - x_t\|^2 + \langle \Delta_t, y_t - x \rangle + \frac{2L_t - L}{2L_t^2}\|\delta_t\|^2.$$

**Proposition 1.** *Assume that for each $t \geq 0$, $\|\Delta_t\|_* \leq \sigma$ and $L_t > L$, then*

$$\phi(y_t) - \phi(x) + \frac{L_t \alpha_t^2 + \mu \alpha_t}{2} \|x - z_t\|^2$$
$$\leq (1 - \alpha_t)[\phi(y_{t-1}) - \phi(x)] + \frac{L_t \alpha_t^2}{2} \|x - z_{t-1}\|^2 + \frac{\sigma^2}{2(L_t - L)} + \alpha_t \langle \Delta_t, x - z_{t-1} \rangle. \tag{8}$$

*Proof.* Define $V_t(x) = \langle \delta_t, x - x_t \rangle + \frac{\mu}{2} \|x - x_t\|^2 + \frac{L_t \alpha_t}{2} \|x - z_{t-1}\|^2$. It is easy to see that $z_t = \arg\min_{x \in \mathbb{R}^d} V_t(x)$. Moreover, notice that $V_t(x)$ is $(L_t \alpha_t + \mu)$-strongly convex. Hence on applying Lemmas 2 and 3, we obtain that for any $x$,

$$V_t(z_t) \leq V_t(x) - \frac{L_t \alpha_t + \mu}{2} \|x - z_t\|^2$$
$$= \langle \delta_t, x - x_t \rangle + \frac{\mu}{2} \|x - x_t\|^2 + \frac{L_t \alpha_t}{2} \|x - z_{t-1}\|^2 - \frac{L_t \alpha_t + \mu}{2} \|x - z_t\|^2$$
$$\leq \phi(x) - \phi(y_t) - \frac{2L_t - L}{2L_t^2} \|\delta_t\|^2 + \frac{L_t \alpha_t}{2} \|x - z_{t-1}\|^2 - \frac{L_t \alpha_t + \mu}{2} \|x - z_t\|^2 + \langle \Delta_t, x - y_t \rangle.$$

Then, $\phi(y_t)$ can be bounded from above, as:

$$\phi(y_t) \leq \phi(x) + \langle \delta_t, x_t - z_t \rangle - \frac{2L_t - L}{2L_t^2} \|\delta_t\|^2 - \frac{L_t \alpha_t}{2} \|z_t - z_{t-1}\|^2$$
$$+ \frac{L_t \alpha_t}{2} \|x - z_{t-1}\|^2 - \frac{L_t \alpha_t + \mu}{2} \|x - z_t\|^2 + \langle \Delta_t, x - y_t \rangle, \tag{9}$$

where the non-positive term $-\frac{\mu}{2} \|z_t - x_t\|^2$ has been dropped from its right-hand-side. On the other hand, by applying Lemma 3 with $x = y_{t-1}$, we get

$$\phi(y_t) - \phi(y_{t-1}) \leq \langle \delta_t, x_t - y_{t-1} \rangle + \langle \Delta_t, y_{t-1} - y_t \rangle - \frac{2L_t - L}{2L_t^2} \|\delta_t\|^2, \tag{10}$$

where the non-positive term $-\frac{\mu}{2} \|y_{t-1} - x_t\|^2$ has also been dropped from the right-hand-side. On multiplying (9) by $\alpha_t$ and (10) by $1 - \alpha_t$, and then adding them together, we obtain

$$\phi(y_t) - \phi(x) \leq (1 - \alpha_t)[\phi(y_{t-1}) - \phi(x)] - \frac{2L_t - L}{2L_t^2} \|\delta_t\|^2 + \mathcal{A} + \mathcal{B} + \mathcal{C} - \frac{L_t \alpha_t^2}{2} \|z_t - z_{t-1}\|^2, \tag{11}$$

where $\mathcal{A} = \langle \delta_t, \alpha_t(x_t - z_t) + (1 - \alpha_t)(x_t - y_{t-1}) \rangle$, $\mathcal{B} = \alpha_t \langle \Delta_t, x - y_t \rangle + (1 - \alpha_t) \langle \Delta_t, y_{t-1} - y_t \rangle$, and $\mathcal{C} = \frac{L_t \alpha_t^2}{2} \|x - z_{t-1}\|^2 - \frac{L_t \alpha_t^2 + \mu \alpha_t}{2} \|x - z_t\|^2$. In the following, we consider to upper bound $\mathcal{A}$ and $\mathcal{B}$. First, by using the update rule of $x_t$ in Algorithm 1 and the Young's inequality[1], we have

$$\mathcal{A} = \langle \delta_t, \alpha_t(x_t - z_{t-1}) + (1 - \alpha_t)(x_t - y_{t-1}) \rangle + \alpha_t \langle \delta_t, z_{t-1} - z_t \rangle$$
$$= \alpha_t \langle \delta_t, z_{t-1} - z_t \rangle \leq \frac{L_t \alpha_t^2}{2} \|z_t - z_{t-1}\|^2 + \frac{\|\delta_t\|^2}{2L_t}. \tag{12}$$

On the other hand, $\mathcal{B}$ can be bounded as

$$\mathcal{B} = \langle \Delta_t, \alpha_t x + (1 - \alpha_t) y_{t-1} - x_t \rangle + \langle \Delta_t, x_t - y_t \rangle = \alpha_t \langle \Delta_t, x - z_{t-1} \rangle + \frac{\langle \Delta_t, \delta_t \rangle}{L_t}$$
$$\leq \alpha_t \langle \Delta_t, x - z_{t-1} \rangle + \frac{\sigma \|\delta_t\|}{L_t}, \tag{13}$$

where the second equality is due to the update rule of $x_t$, and the last step is from the Cauchy-Schwartz inequality and the boundedness of $\Delta_t$. Hence, plugging (12) and (13) into (11),

$$\phi(y_t) - \phi(x) \leq (1 - \alpha_t)[\phi(y_{t-1}) - \phi(x)] - \frac{(L_t - L) \|\delta_t\|^2}{2L_t^2} + \frac{\sigma \|\delta_t\|}{L_t} + \alpha_t \langle \Delta_t, x - z_{t-1} \rangle + \mathcal{C}$$
$$\leq (1 - \alpha_t)[\phi(y_{t-1}) - \phi(x)] + \frac{\sigma^2}{2(L_t - L)} + \alpha_t \langle \Delta_t, x - z_{t-1} \rangle + \mathcal{C},$$

where the last step is due to the fact that $-ax^2 + bx \leq \frac{b^2}{4a}$ with $a, b > 0$. On re-arranging terms, we obtain (8). $\qquad \square$

Let the optimal solution in problem (6) be $x^*$. From the update rules in Algorithm 1, we observe that the triplet $(x_t, y_{t-1}, z_{t-1})$ depends on the random process $\xi_{[t-1]} \equiv \{\xi_0, \ldots, \xi_{t-1}\}$ and hence is also random. Clearly, $z_{t-1}$ and $x^*$ are independent of $\xi_t$. Thus,

$$
\begin{aligned}
\mathbb{E}_{\xi_{[t]}} \langle \Delta_t, x^* - z_{t-1} \rangle &= \mathbb{E}_{\xi_{[t-1]}} \mathbb{E}_{\xi_{[t]}} [\langle \Delta_t, x^* - z_{t-1} \rangle | \xi_{[t-1]}] = \mathbb{E}_{\xi_{[t-1]}} \mathbb{E}_{\xi_t} [\langle \Delta_t, x^* - z_{t-1} \rangle] \\
&= \mathbb{E}_{\xi_{[t-1]}} \langle x^* - z_{t-1}, \mathbb{E}_{\xi_t}[\Delta_t] \rangle = 0,
\end{aligned}
$$

where the first equality uses $\mathbb{E}_x[h(x)] = \mathbb{E}_y \mathbb{E}_x[h(x)|y]$, and the last equality is from our assumption that the stochastic gradient $G(x, \xi)$ is unbiased. Taking expectations on both sides of (8) with $x = x^*$, we obtain the following corollary, which will be useful in proving the subsequent theorems.

**Corollary 1.**

$$
\mathbb{E}[\phi(y_t)] - \phi(x^*) + \frac{L_t \alpha_t^2 + \mu \alpha_t}{2} \mathbb{E}[\|x^* - z_t\|^2]
$$
$$
\leq (1 - \alpha_t)(\mathbb{E}[\phi(y_{t-1})] - \phi(x^*)) + \frac{L_t \alpha_t^2}{2} \mathbb{E}[\|x^* - z_{t-1}\|^2] + \frac{\sigma^2}{2(L_t - L)}.
$$

So far, the choice of $L_t$ and $\alpha_t$ in Algorithm 1 has been left unspecified. In the following, we will show that with a good choice of $L_t$ and $\alpha_t$, (the expectation of) $\phi(y_t)$ converges rapidly to $\phi(x^*)$.

**Theorem 1.** *Assume that $\mathbb{E}[\|x^* - z_t\|^2] \leq D^2$ for some $D$. Set*

$$
L_t = b(t+1)^{\frac{3}{2}} + L, \quad \alpha_t = \frac{2}{t+2}, \tag{14}
$$

*where $b > 0$ is a constant. Then the expected error of Algorithm 1 can be bounded as*

$$
\mathbb{E}[\phi(y_N)] - \phi(x^*) \leq \frac{3D^2 L}{N^2} + \left( 3D^2 b + \frac{5\sigma^2}{3b} \right) \frac{1}{\sqrt{N}}. \tag{15}
$$

If $\sigma$ were known, we can set $b$ to the optimal choice of $\frac{\sqrt{5}\sigma}{3D}$, and the bound in (15) becomes $\frac{3D^2 L}{N^2} + \frac{2\sqrt{5}\sigma D}{\sqrt{N}}$.

Note that so far $\phi(x)$ is only assumed to be convex. As is shown in the following theorem, the convergence rate can be further improved by assuming strong convexity. This also requires another setting of $\alpha_t$ and $L_t$ which is different from that in (14).

**Theorem 2.** *Assume the same conditions as in Theorem 1, except that $\phi(x)$ is $\mu$-strongly convex. Set*

$$
L_t = L + \mu \lambda_{t-1}^{-1}, \; for \; t \geq 1; \quad \alpha_t = \sqrt{\lambda_{t-1} + \frac{\lambda_{t-1}^2}{4}} - \frac{\lambda_{t-1}}{2}, \; for \; t \geq 1, \tag{16}
$$

*where $\lambda_t \equiv \Pi_{k=1}^t (1 - \alpha_t)$ for $t \geq 1$ and $\lambda_0 = 1$. Then, the expected error of Algorithm 1 can be bounded as*

$$
\mathbb{E}[\phi(y_N)] - \phi(x^*) \leq \frac{2(L + \mu)D^2}{N^2} + \frac{6\sigma^2}{N\mu}. \tag{17}
$$

In comparison, FOLOS only converges as $\mathcal{O}(\log(N)/N)$ for strongly convex objectives.

### 3.2 Remarks

As in recent studies on stochastic composite optimization [13], the error bounds in (15) and (17) consist of two terms: a faster term which is related to the smooth component and a slower term related to the non-smooth component. SAGE benefits from using the structure of the problem and accelerates the convergence of the smooth component. On the other hand, many stochastic (sub)gradient-based algorithms like FOLOS do not separate the smooth from the non-smooth part, but simply treat the whole objective as non-smooth. Consequently, convergence of the smooth component is also slowed down to $\mathcal{O}(1/\sqrt{N})$.

As can be seen from (15) and (17), the convergence of SAGE is essentially encumbered by the variance of the stochastic subgradient. Recall that the variance of the average of $p$ i.i.d. random

variables is equal to $1/p$ of the original variance. Hence, as in Pegasos [1], $\sigma$ can be reduced by estimating the subgradient from a data subset.

Unlike the AC-SA algorithm in [13], the settings of $L_t$ and $\alpha_t$ in (14) do not require knowledge of $\sigma$ and the number of iterations, both of which can be difficult to estimate in practice. Moreover, with the use of a sparsity-promoting $\psi(x)$, SAGE can produce a sparse solution (as will be experimentally demonstrated in Section 5) while AC-SA cannot. This is because in SAGE, the output $y_t$ is obtained from a generalized gradient update. With a sparsity-promoting $\psi(x)$, this reduces to a (soft) thresholding step, and thus ensures a sparse solution. On the other hand, in each iteration of AC-SA, its output is a convex combination of two other variables. Unfortunately, adding two vectors is unlikely to produce a sparse vector.

### 3.3 Efficient Computation of $y_t$

The computational efficiency of Algorithm 1 hinges on the efficient computation of $y_t$. Recall that $y_t$ is just the generalized gradient update, and so is not significantly more expensive than the gradient update in traditional algorithms. Indeed, the generalized gradient update is often a central component in various optimization and machine learning algorithms. In particular, Duchi and Singer [3] showed how this can be efficiently computed with the various smooth and non-smooth regularizers, including the $\ell_1, \ell_2, \ell_2^2, \ell_\infty$, Berhu and matrix norms. Interested readers are referred to [3] for details.

## 4   Accelerated Gradient Method for Online Learning

In this section, we extend the proposed accelerated gradient scheme for online learning of (2). The algorithm, shown in Algorithm 2, is similar to the stochastic version in Algorithm 1.

---

**Algorithm 2** SAGE-based Online Learning Algorithm.

Inputs: Sequences $\{L_t\}$ and $\{\alpha_t\}$, where $L_t > L$ and $0 < \alpha_t < 1$.
Initialize: $z_1 = y_1$.
**loop**
  $x_t = (1 - \alpha_t)y_{t-1} + \alpha_t z_{t-1}$.
  Output $y_t = \arg\min_x \left\{ \langle \nabla f_{t-1}(x_t), x - x_t \rangle + \frac{L_t}{2}\|x - x_t\|^2 + \psi(x) \right\}$.
  $z_t = z_{t-1} - \alpha_t(L_t + \mu\alpha_t)^{-1}[L_t(x_t - y_t) + \mu(z_{t-1} - x_t)]$.
**end loop**

---

First, we introduce the following lemma, which plays a similar role as its stochastic counterpart of Lemma 3. Moreover, let $\delta_t \equiv L_t(x_t - y_t)$ be the gradient mapping related to the updating of $y_t$.

**Lemma 4.** *For $t > 1$, $\phi_t(x)$ can be quadratically bounded from below as*

$$\phi_{t-1}(x) \geq \phi_{t-1}(y_t) + \langle \delta_t, x - x_t \rangle + \frac{\mu}{2}\|x - x_t\|^2 + \frac{2L_t - L}{2L_t^2}\|\delta_t\|^2.$$

**Proposition 2.** *For any $x$ and $t \geq 1$, assume that there exists a subgradient $\hat{g}(x) \in \partial\psi(x)$ such that $\|\nabla f_t(x) + \hat{g}(x)\|_* \leq Q$. Then for Algorithm 2,*

$$\phi_{t-1}(y_{t-1}) - \phi_{t-1}(x) \leq \frac{Q^2}{2(1-\alpha_t)(L_t - L)} + \frac{L_t}{2\alpha_t}\|x - z_{t-1}\|^2 - \frac{L_t + \mu\alpha_t}{2\alpha_t}\|x - z_t\|^2$$
$$+ \frac{(1-\alpha_t^2)L_t - \alpha_t(1-\alpha_t)L}{2}\|y_{t-1} - z_{t-1}\|^2 - \frac{L_t}{2}\|z_t - y_t\|^2. \tag{18}$$

*Proof Sketch.* Define $\tau_t = L_t\alpha_t^{-1}$. From the update rule of $z_t$, one can check that

$$z_t = \arg\min_x V_t(x) \equiv \langle \delta_t, x - x_t \rangle + \frac{\mu}{2}\|x - x_t\|^2 + \frac{\tau_t}{2}\|x - z_{t-1}\|^2.$$

Similar to the analysis in obtaining (9), we can obtain

$$\phi_{t-1}(y_t) - \phi_{t-1}(x) \leq \langle \delta_t, x_t - z_t \rangle - \frac{2L_t - L}{2L_t^2}\|\delta_t\|^2 - \frac{\tau_t}{2}\|z_t - z_{t-1}\|^2 + \frac{\tau_t}{2}\|x - z_{t-1}\|^2 - \frac{\tau_t + \mu}{2}\|x - z_t\|^2. \tag{19}$$

On the other hand,

$$\langle \delta_t, x_t - z_t \rangle - \frac{\|\delta_t\|^2}{2L_t} = \frac{L_t}{2}(\|z_t - x_t\|^2 - \|z_t - y_t\|^2)$$

$$\leq \frac{L_t}{2\alpha_t}\|z_t - z_{t-1}\|^2 + \frac{L_t(1-\alpha_t)}{2}\|z_{t-1} - y_{t-1}\|^2 - \frac{L_t}{2}\|z_t - y_t\|^2, \quad (20)$$

on using the convexity of $\|\cdot\|^2$. Using (20), the inequality (19) becomes

$$\phi_{t-1}(y_t) - \phi_{t-1}(x) \leq \frac{L_t(1-\alpha_t)}{2}\|z_{t-1} - y_{t-1}\|^2 - \frac{L_t}{2}\|z_t - y_t\|^2$$

$$- \frac{L_t - L}{2L_t^2}\|\delta_t\|^2 + \frac{\tau_t}{2}\|x - z_{t-1}\|^2 - \frac{\tau_t + \mu}{2}\|x - z_t\|^2. \quad (21)$$

On the other hand, by the convexity of $\phi_{t-1}(x)$ and the Young's inequality, we have

$$\phi_{t-1}(y_{t-1}) - \phi_{t-1}(y_t) \leq \langle \nabla f_{t-1}(y_{t-1}) + \hat{g}_{t-1}(y_{t-1}), y_{t-1} - y_t \rangle$$

$$\leq \frac{Q^2}{2(1-\alpha_t)(L_t - L)} + \frac{(1-\alpha_t)(L_t - L)}{2}\|y_{t-1} - y_t\|^2. \quad (22)$$

Moreover, by using the update rule of $x_t$ and the convexity of $\|\cdot\|^2$, we have

$$\|y_{t-1} - y_t\|^2 = \|(y_{t-1} - x_t) + (x_t - y_t)\|^2 = \|\alpha_t(y_{t-1} - z_{t-1}) + (x_t - y_t)\|^2$$

$$\leq \alpha_t \|y_{t-1} - z_{t-1}\|^2 + (1-\alpha_t)^{-1}\|x_t - y_t\|^2 = \alpha_t \|y_{t-1} - z_{t-1}\|^2 + \frac{\|\delta_t\|^2}{(1-\alpha_t)L_t^2}. \quad (23)$$

On using (23), it follows from (22) that

$$\phi_{t-1}(y_{t-1}) - \phi_{t-1}(y_t) \leq \frac{Q^2}{2(1-\alpha_t)(L_t-L)} + \frac{\alpha_t(1-\alpha_t)(L_t-L)}{2}\|y_{t-1}-z_{t-1}\|^2 + \frac{L_t-L}{2L_t^2}\|\delta_t\|^2.$$

Inequality (18) then follows immediately by adding this to (21). $\qquad\square$

**Theorem 3.** *Assume that $\mu = 0$, and $\|x^* - z_t\| \leq D$ for $t \geq 1$. Set $\alpha_t = a$ and $L_t = aL\sqrt{t-1} + L$, where $a \in (0,1)$ is a constant. Then the regret of Algorithm 2 can be bounded as*

$$\sum_{t=1}^{N}[\phi_t(y_t) - \phi_t(x^*)] \leq \frac{LD^2}{2a} + \left[\frac{LD^2}{2} + \frac{Q^2}{a(1-a)L}\right]\sqrt{N}.$$

**Theorem 4.** *Assume that $\mu > 0$, and $\|x^* - z_t\| \leq D$ for $t \geq 1$. Set $\alpha_t = a$, and $L_t = a\mu t + L + a^{-1}(\mu - L)_+$, where $a \in (0,1)$ is a constant. Then the regret of Algorithm 2 can be bounded as*

$$\sum_{t=1}^{N}[\phi_t(y_t) - \phi_t(x^*)] \leq \left[\frac{(2a + a^{-1})\mu + L}{2a}\right]D^2 + \frac{Q^2}{2a(1-a)\mu}\log(N+1).$$

In particular, with $a = \frac{1}{2}$, the regret bound reduces to $\left(\frac{3\mu}{2} + L\right)D^2 + \frac{2Q^2}{\mu}\log(N+1)$.

## 5 Experiments

In this section, we perform experiments on the stochastic optimization of (2). Two data sets are used[2] (Table 1). The first one is the pcmac data set, which is a subset of the 20-newsgroup data set from [18], while the second one is the RCV1 data set, which is a filtered collection of the Reuters RCV1 from [19]. We choose the square loss for $\ell(\cdot,\cdot)$ and the $\ell_1$ regularizer for $\Omega(\cdot)$ in (2). As discussed in Section 3.3 and [3], the generalized gradient update can be efficiently computed by soft thresholding in this case. Moreover, we do not use strong convexity and so $\mu = 0$.

We compare the proposed SAGE algorithm (with $L_t$ and $\alpha_t$ in (14)) with three recent algorithms: (1) FOLOS [3]; (2) SMIDAS [4]; and (3) SCD [4]. For fair comparison, we compare their convergence

behavior w.r.t. both the number of iterations and the number of data access operations, the latter of which has been advocated in [4] as an implementation-independent measure of time. Moreover, the efficiency tricks for sparse data described in [4] are also implemented. Following [4], we set the regularization parameter $\lambda$ in (2) to $10^{-6}$. The $\eta$ parameter in SMIDAS is searched over the range of $\{10^{-6}, 10^{-5}, 10^{-4}, 10^{-3}, 10^{-2}, 10^{-1}\}$, and the one with the lowest $\ell_1$-regularized loss is used. As in Pegasos [1], the (sub)gradient is computed from small sample subsets. The subset size $p$ is set to $\min(0.01m, 500)$, where $m$ is the data set size. This is used on all the algorithms except SCD, since SCD is based on coordinate descent and is quite different from the other stochastic subgradient algorithms.[3] All the algorithms are trained with the same maximum amount of "time" (i.e., number of data access operations).

Table 1: Summary of the data sets.

| data set | #features | #instances | sparsity |
|---|---|---|---|
| pcmac | 7,511 | 1,946 | 0.73% |
| RCV1 | 47,236 | 193,844 | 0.12% |

Results are shown in Figure 1. As can be seen, SAGE requires much fewer iterations for convergence than the others (Figures 1(a) and 1(e)). Moreover, the additional costs on maintaining $x_t$ and $z_t$ are small, and the most expensive step in each SAGE iteration is in computing the generalized gradient update. Hence, its per-iteration complexity is comparable with the other (sub)gradient schemes, and its convergence in terms of the number of data access operations is still the fastest (Figures 1(b), 1(c), 1(f) and 1(g)). Moreover, the sparsity of the SAGE solution is comparable with those of the other algorithms (Figures 1(d) and 1(h)).

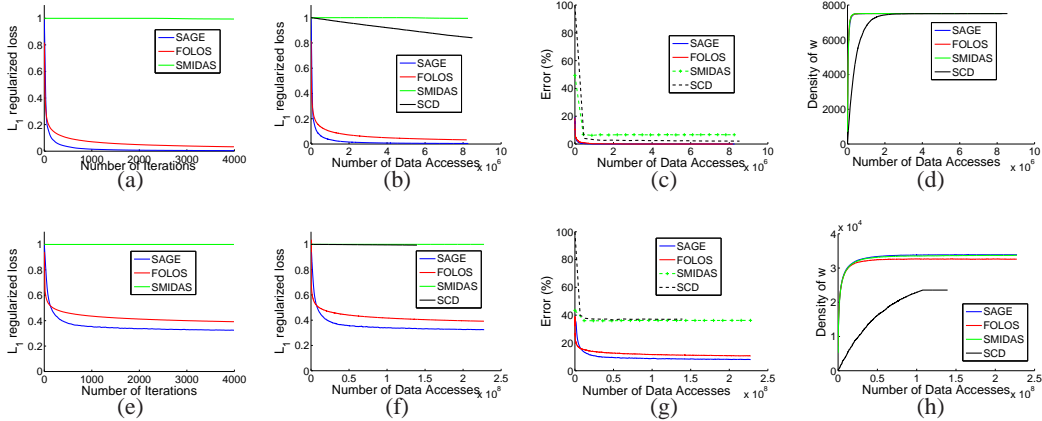

Figure 1: Performance of the various algorithms on the pcmac (upper) and RCV1 (below) data sets.

## 6 Conclusion

In this paper, we developed a novel accelerated gradient method (SAGE) for stochastic convex composite optimization. It enjoys the computational simplicity and scalability of traditional (sub)gradient methods but are much faster, both theoretically and empirically. Experimental results show that SAGE outperforms recent (sub)gradient descent methods. Moreover, SAGE can also be extended to online learning, obtaining the best regret bounds currently known.

## Acknowledgment

This research has been partially supported by the Research Grants Council of the Hong Kong Special Administrative Region under grant 615209.

## Footnotes

[1] The Young's inequality states that $\langle x, y \rangle \leq \frac{\|x\|^2}{2a} + \frac{a\|y\|^2}{2}$ for any $a > 0$.

[2]Downloaded from http://people.cs.uchicago.edu/~vikass/svmlin.html and http://www.cs.ucsb.edu/~wychen/sc.html.

[3]For the same reason, an SCD iteration is also very different from an iteration in the other algorithms. Hence, SCD is not shown in the plots on the regularized loss versus number of iterations.

# References

[1] S. Shalev-Shwartz, Y. Singer, and N. Srebro. Pegasos: Primal estimated sub-gradient solver for SVM. In *Proceedings of the 24th International Conference on Machine Learning*, pages 807–814, Corvalis, Oregon, USA, 2007.

[2] A. Bordes, L. Bottou, and P. Gallinari. SGD-QN: Careful Quasi-Newton Stochastic Gradient Descent. *Journal of Machine Learning Research*, 10:1737–1754, 2009.

[3] J. Duchi and Y. Singer. Online and batch learning using forward looking subgradients. Technical report, 2009.

[4] S. Shalev-Shwartz and A. Tewari. Stochastic methods for $\ell_1$ regularized loss minimization. In *Proceedings of the 26th International Conference on Machine Learning*, pages 929–936, Montreal, Quebec, Canada, 2009.

[5] L. Bottou and O. Bousquet. The tradeoffs of large scale learning. In *Advances in Neural Information Processing Systems 20*. 2008.

[6] S. Shalev-Shwartz and N. Srebro. SVM optimization: Inverse dependence on training set size. In *Proceedings of the 25th International Conference on Machine Learning*, pages 928–935, Helsinki, Finland, 2008.

[7] Y. Nesterov. A method for unconstrained convex minimization problem with the rate of convergence $o(\frac{1}{k^2})$. *Doklady AN SSSR (translated as Soviet. Math. Docl.)*, 269:543–547, 1983.

[8] Y. Nesterov. Gradient methods for minimizing composite objective function. CORE Discussion Paper 2007/76, Catholic University of Louvain, September 2007.

[9] A. Beck and M. Teboulle. A fast iterative shrinkage-thresholding algorithm for linear inverse problems. *SIAM Journal on Imaging Sciences*, 2:183–202, 2009.

[10] R. Tibshirani. Regression shrinkage and selection via the Lasso. *Journal of the Royal Statistical Society: Series B*, 58:267–288, 1996.

[11] S. Ji, L. Sun, R. Jin, and J. Ye. Multi-label multiple kernel learning. In *Advances in Neural Information Processing Systems 21*. 2009.

[12] S. Ji and J. Ye. An accelerated gradient method for trace norm minimization. In *Proceedings of the International Conference on Machine Learning*. Montreal, Canada, 2009.

[13] G. Lan. An optimal method for stochastic composite optimization. Technical report, School of Industrial and Systems Engineering, Georgia Institute of Technology, 2009.

[14] Y. Nesterov and I.U.E. Nesterov. *Introductory Lectures on Convex Optimization: A Basic Course*. Kluwer, 2003.

[15] S.M. Kakade and S. Shalev-Shwartz. Mind the duality gap: Logarithmic regret algorithms for online optimization. In *Advances in Neural Information Processing Systems 21*. 2009.

[16] A. Beck and M. Teboulle. Mirror descent and nonlinear projected subgradient methods for convex optimization. *Operations Research Letters*, 31(3):167–175, 2003.

[17] S.J. Wright, R.D. Nowak, and M.A.T. Figueiredo. Sparse reconstruction by separable approximation. In *Proceedings of the International Conference on Acoustics, Speech, and Signal Processing*, Las Vegas, Nevada, USA, March 2008.

[18] V. Sindhwani and S.S. Keerthi. Large scale semi-supervised linear SVMs. In *Proceedings of the SIGIR Conference on Research and Development in Information Retrieval*, pages 477–484, Seattle, WA, USA, 2006.

[19] Y. Song, W.Y. Chen, H. Bai, C.J. Lin, and E.Y. Chang. Parallel spectral clustering. In *Proceedings of the European Conference on Machine Learning*, pages 374–389, Antwerp, Belgium, 2008.

